# Learning curves for stochastic gradient descent in linear feedforward networks

**Justin Werfel**
Dept. of EECS
MIT
Cambridge, MA 02139
jkwerfel@mit.edu

**Xiaohui Xie**
Dept. of Molecular Biology
Princeton University
Princeton, NJ 08544
xhx@princeton.edu

**H. Sebastian Seung**
HHMI
Dept. of Brain & Cog. Sci.
MIT
Cambridge, MA 02139
seung@mit.edu

## Abstract

Gradient-following learning methods can encounter problems of implementation in many applications, and stochastic variants are frequently used to overcome these difficulties. We derive quantitative learning curves for three online training methods used with a linear perceptron: direct gradient descent, node perturbation, and weight perturbation. The maximum learning rate for the stochastic methods scales inversely with the first power of the dimensionality of the noise injected into the system; with sufficiently small learning rate, all three methods give identical learning curves. These results suggest guidelines for when these stochastic methods will be limited in their utility, and considerations for architectures in which they will be effective.

## 1 Introduction

Learning in artificial systems can be formulated as optimization of an objective function which quantifies the system's performance. A typical approach to this optimization is to follow the gradient of the objective function with respect to the tunable parameters of the system. Frequently this is accomplished directly, by calculating the gradient explicitly and updating the parameters by a small step in the direction of locally greatest improvement.

In many circumstances, however, attempts at direct gradient-following can encounter problems. In VLSI and other hardware implementations, computation of the gradient may be excessively unwieldy, if not impossible due to unavoidable imperfections in manufacturing [1]-[5]. In some cases, as with many where the reinforcement learning framework is used, there may be no explicit form for the objective function and hence no way of calculating its gradient [6]. And in biological systems, any argument that direct gradient calculation might be what the system is actually doing typically encounters severe obstacles. For instance, backpropagation, the standard method for training artificial neural networks, requires two-way, multipurpose synapses, units with global knowledge about the system that are able to recognize different kinds of signals and treat them in very different ways, and (in the case of trajectory learning) the ability to run backwards in time, all of which strain the bounds of biological plausibility [1, 7]. For reasons such as these, there has been broad interest in stochastic methods which approximate the gradient on average.

Compared to a method that follows the true gradient directly, we would intuitively expect a stochastic gradient-following approach to learn more slowly. The stochastic algorithms in this study use a reinforcement-learning framework with a single reward signal, which is assigned based on the contributions of all the tunable parameters of the system; that single reward is all that is available to evaluate how every one of the parameters should be updated, in contrast to a true-gradient method where the optimal updates are all specified. Moreover, if the network is made larger and the number of parameters thereby increased, this credit assignment problem becomes still more difficult; thus we expect the performance of stochastic gradient methods to scale up with network size more poorly than deterministic methods. However, under some circumstances stochastic methods can be equally as effective as direct ones in training even large networks, generating near-identical learning curves (see, e.g., Fig. 2 below). Under what circumstances, then, will stochastic gradient descent have performance comparable to that of the deterministic variety? And how good can that performance be?

In this paper, we investigate these issues quantitatively by calculating the learning curves for a linear perceptron using a direct gradient method and two stochastic methods, node perturbation and weight perturbation. We find that the maximum learning speed for each algorithm scales inversely with the first power of the dimensionality of the noise injected into the system; this result is in contradiction to previous work, which reported maximum learning speed scaling inversely with the square root of the dimensionality of the injected noise [4]. Additionally, when learning rates are chosen to be very low, and such that the weight updates prescribed by each method are equal on average, we find that all three methods give identical learning curves.

## 2   Perceptron comparison

Direct and stochastic gradient approaches are general classes of training methods. We study the operation of exemplars of both on a feedforward linear perceptron, which has the advantage over the nonlinear case that the learning curves can be calculated exactly [8]. We have $N$ input units and $M$ output units, connected by a weight matrix $w$ of $MN$ elements; outputs in response to an input $x$ are given by $y = wx$. For the ensemble of possible inputs, we want to train the network to produce desired corresponding outputs $y = d$; in order to ensure that this task is realizable by the network, we assume the existence of a teacher network $w^*$ such that $d = w^*x$. We use the squared error function

$$E = \frac{1}{2}|y - d|^2 = \frac{1}{2}|(w - w^*)x|^2 = \frac{1}{2}|Wx|^2 \tag{1}$$

where we have defined the matrix $W \equiv w - w^*$. We train the network with an online approach, choosing at each time step an input vector $x$ with components drawn from a Gaussian distribution with mean 0 and unit variance, and using it to construct a weight update according to one of the three prescriptions below.

The online gradient-following approach explicitly uses the gradient of the error function for a given input to determine the weight update:

$$\Delta W_{\text{OL}} = -\eta \nabla E$$

where $\eta > 0$ is the learning rate. This is the approach taken, e.g., by backpropagation.

In the stochastic algorithms, the gradient is not calculated directly; instead, some noise is introduced into the system, affecting its error for a given input, and the difference between the error with and without noise is used to estimate the gradient. The simplest case is when noise is added directly to the weight matrix:

$$E'_{\text{WP}} = \frac{1}{2}|(W + \psi)x|^2$$

Such an approach is sometimes termed 'weight perturbation' [2, 4]. We choose each element of the noise matrix $\psi$ from a Gaussian distribution with mean 0 and variance $\sigma^2$. Intuitively, if the addition of the noise lowers the error, that perturbation to the weight matrix is retained, which will mean lower error for that input in future. Conversely, if the noise leads to an increase in error, the opposite change is made to the weights; the effect of small noise on error can be approximated as linear, and the opposite change in weights will lead to the opposite change in error, again decreasing error for that input in future. These two cases can be combined into the single weight update

$$\Delta W_{\mathrm{WP}} = -\frac{\eta}{\sigma^2}(E'_{\mathrm{WP}} - E)\psi$$

A more subtle way to introduce stochasticity involves adding the noise to the output of each output unit rather than to every weight:

$$E'_{\mathrm{NP}} = \frac{1}{2}|Wx + \xi|^2$$

Such an approach is sometimes called 'node perturbation' [1, 3]. Here if the noise leads to a decrease in error, the weights are adjusted in such a way as to move the outputs in the direction of that noise. The degree of freedom for each output unit corresponds to the adjustment of its threshold, making the unit more or less responsive to a given pattern of input activity. The elements of $\xi$ are again chosen independently from a Gaussian distribution with variance $\sigma^2$; here $\xi$ has $M$ elements, whereas $\psi$ in the previous case had $MN$. The REINFORCE framework [9] gives for the weight update

$$\Delta W_{\mathrm{NP}} = -\frac{\eta}{\sigma^2}(E'_{\mathrm{NP}} - E)\xi x^T$$

These stochastic frameworks produce weight updates identical to that of direct gradient descent on the error function when averaged over all values of the noise [4, 9], which is the sense in which they constitute stochastic gradient descent. This result is easy to verify in the particular forms taken by $\Delta W_{\mathrm{NP}}$ and $\Delta W_{\mathrm{WP}}$ here, shown below.

## 2.1 Online gradient method

Taking the gradient of the error function of Eq. 1 gives

$$\Delta W_{\mathrm{OL}} = -\eta W x x^T \tag{2}$$

as the individual weight update for particular values of $W$ and $x$. This rule lets us calculate a recursion relation specifying how $\|W\|^2$ changes from one time step to the next:

$$\sum_{ij}\langle(W_{ij}^{(t)})^2\rangle_t = (1 - 2\eta + (N + 2)\eta^2)\sum_{ij}\left(W_{ij}^{(t-1)}\right)^2 \tag{3}$$

where the parenthesized superscript is a time index, and the subscripted angle brackets denote an average over the ensemble of all inputs at that time. Applying this recursion relation gives an expression for the average error as a function of time, where the unsubscripted brackets indicate a mean taken over all inputs at every time step:

$$\langle E_{\mathrm{OL}}^{(t)}\rangle = (1 - 2\eta + (N + 2)\eta^2)^t E^{(0)}$$

In a single online learning run, $E^{(t)}$ would depend on the particular values of $x$ that were randomly chosen; averaging over the ensemble of possible inputs $x$ removes this variation. We therefore use this averaged error $\langle E^{(t)}\rangle$ as the learning curve measuring the performance of the system.

We have the condition for convergence of the average error

$$\eta < \frac{2}{N + 2}$$

The limit on $\eta$ has this dependence on $N$ because of the randomness inherent in an online training regimen; the exact gradient for error due to a given single input $x$ will not in general match that for error averaged over the entire ensemble of inputs. We can write an expression for the $ij$-component of the weight update, explicitly in terms of 'gradient signal' (term multiplying $W_{ij}$) plus 'gradient noise' [1] (contamination from other components of $W$ due to projection onto $x$):

$$\Delta W_{ij} = -\eta \left( W_{ij} x_j^2 + \sum_{k \neq j} W_{ik} x_k x_j \right)$$

We can similarly rewrite Eq. 3 as

$$\sum_{ij} \langle W_{ij}^{(1)2} \rangle = \sum_{ij} W_{ij}^{(0)2} (1 - 2\eta + 3\eta^2) + \eta^2 (N-1) \sum_{ij} W_{ij}^{(0)}$$

where the first term is due entirely to the gradient signal and the second to the gradient noise; choosing $\eta \lesssim 1/N$ allows the signal to be revealed via averaging over $\gtrsim N$ samples (see also the Discussion). This gradient noise is common to all three algorithms considered here.

## 2.2  Node perturbation

Here averages are taken at each step not only over the inputs $x$ but also over the noise $\xi$. The weight update, recursion relation, learning curve, and convergence condition are

$$\Delta W_{\mathrm{NP}} = -\frac{\eta}{\sigma^2}(\xi^T W x + \frac{1}{2}\xi^T \xi)\xi x^T$$

$$\sum_{ij} \langle W_{ij}^{(t)2} \rangle_t = \sum_{ij} W_{ij}^{(t-1)2}(1 - 2\eta + \eta^2(M+2)(N+2))$$

$$+ \frac{1}{4}\eta^2 \sigma^2 M N (M+2)(M+4))$$

$$\langle E_{\mathrm{NP}}^{(t)} \rangle = \left( E^{(0)} - \frac{\eta \sigma^2 (M+2)(M+4)MN/8}{2 - (N+2)(M+2)\eta} \right)(1 - 2\eta + (M+2)(N+2)\eta^2)^t$$

$$+ \frac{\eta \sigma^2 (M+2)(M+4)MN/8}{2 - (M+2)(N+2)\eta}$$

$$\eta < \frac{2}{(M+2)(N+2)}$$

In this case the recursion relation has not only a multiplicative term as before but also an additive one. The latter is a result of the noise $\xi$; when $W$ is far from the minimum of the objective function, $\xi$ will typically be small in comparison to $Wx$ and the additive term will be negligible, but close to the minimum the noise will prevent the system from attaining arbitrarily low error. This effect appears also in the learning curve. The limit on $\eta$ is stricter by a factor of $M$, the dimensionality of the noise, as discussed below.

## 2.3  Weight perturbation

The same approach as before gives in this case

$$\Delta W_{\mathrm{WP}} = -\frac{\eta}{\sigma^2}(x^T \psi^T W x + \frac{1}{2}x^T \psi^T \psi x)\psi$$

$$\sum_{ij} \langle W_{ij}^{(t)2} \rangle_t = \sum_{ij} W_{ij}^{(t-1)2}(1 - 2\eta + \eta^2(MN+2)(N+2))$$

$$+ \frac{1}{4}\eta^2\sigma^2(M^3N^3 + 2M^2N^3 + 2M^3N^2 + 16M^2N^2 + 24MN)$$

$$\langle E_{\text{WP}}^{(t)} \rangle = (E^{(0)} - \frac{\eta\sigma^2 MN(MN(M+2)(N+2) + 12(MN+2))}{8(2 - (N+2)(MN+2)\eta)})$$

$$\cdot (1 - 2\eta + (N+2)(MN+2)\eta^2)^t$$

$$+ \frac{\eta\sigma^2 MN(MN(M+2)(N+2) + 12(MN+2))}{8(2 - (N+2)(MN+2)\eta)}$$

$$\eta < \frac{2}{(MN+2)(N+2)}$$

As with node perturbation, the recursion relation involves both multiplicative and additive terms, and the learning curve shows nonzero residual error even at infinite time. The limit on $\eta$ is a further factor of $N$ smaller, corresponding to the greater dimensionality of $\psi$ compared to $\xi$.

## 3 Comparison of learning curves

All three of the above learning curves $\langle E^{(t)} \rangle$ take the form $\bar{E}(a(\eta))^t + b(\eta, \sigma)$, where $b$ is the residual error which the network will approach as $t \to \infty$ if learning converges, $\bar{E} \equiv E^{(0)} - b$ is the transient error, and $a$ is a multiplicative factor by which $\bar{E}$ changes at each time step. The magnitude of $a$, which depends on the parameter $\eta$ but not on $\sigma$, determines whether the average error will converge and the rate at which it will do so. For the online gradient method, $b = 0$; a network trained this way, if it converges, will approach zero error as $t \to \infty$. The stochastic algorithms have positive residual noise $b$, which depends on both $\eta$ and $\sigma$; in the limit $\sigma \to 0$, this residual error vanishes. Of course, $\sigma$ cannot be set directly to 0 or the stochastic algorithms will cease to function.

### 3.1 Maximal learning rates

The analysis of the previous section suggests at least two reasonable ways to compare these different algorithms with respect to performance. One is to choose the optimal learning rate for each, that value of $\eta$ for which the average error converges most quickly. The learning curves, to highest order in $\eta$, $M$, and $N$, then become

$$\langle E_{\text{OL}}^{(t)} \rangle = \bar{E}\left(1 - \frac{1}{N}\right)^t$$

$$\langle E_{\text{NP}}^{(t)} \rangle = \bar{E}\left(1 - \frac{1}{MN}\right)^t + \frac{1}{8}\sigma^2 M^2$$

$$\langle E_{\text{WP}}^{(t)} \rangle = \bar{E}\left(1 - \frac{1}{MN^2}\right)^t + \frac{1}{8}\sigma^2 M^2 N$$

Direct gradient descent, then, can train a network faster than can node perturbation, which in turn is faster than weight perturbation.

The noise takes different forms in the two stochastic variants. For node perturbation, $\xi_i$ is added directly to the $i$th output unit; for weight perturbation, the quantity added to the same output unit is $\sum_{ij} \psi_{ij} x_j$. By the central limit theorem, the latter approaches a Gaussian with mean 0 and variance $N\sigma^2$ for large $N$. For most direct comparison of the two stochastic variants, therefore, $\sigma$ for $\xi$ should be chosen a factor $\sqrt{N}$ larger than for $\psi$. With this choice, the residual error for the two stochastic variants becomes identical, and the learning curves differ only in their rates of convergence.

### 3.2 Equal average updates

A second way to compare the algorithms is to choose learning rates such that all three have the same average weight update. As noted above, choosing the same value of $\eta$ in all three cases will ensure this condition. That common value of $\eta$ must be small enough that all three algorithms converge; if we take $\eta \ll \frac{1}{MN^2}$, the learning curves become

$$
\begin{aligned}
\langle E_{\text{OL}}^{(t)} \rangle &= \bar{E}(1-2\eta)^t \\
\langle E_{\text{NP}}^{(t)} \rangle &= \bar{E}\left(1-2\eta\right)^t + \frac{1}{16}\eta\sigma^2 M^3 N \\
\langle E_{\text{WP}}^{(t)} \rangle &= \bar{E}\left(1-2\eta\right)^t + \frac{1}{16}\eta\sigma^2 M^3 N^3
\end{aligned}
$$

We began by saying that, because of the credit assignment problem of choosing updates to many parameters based on a single reward signal, intuition is that a stochastic gradient-following approach should learn more slowly than a direct one. However, for equal small $\eta$, the average error for all three algorithms converges at the same rate. Weight perturbation approaches a larger value of residual error than does node perturbation; however, in the $\sigma \to 0$ limit, the residual error vanishes for both.

## 4    Discussion

In a linear feedforward network of $N$ input and $M$ output units, in terms of the maximum possible rate of convergence of average error, online gradient descent on a squared error function is faster by a factor of $M$ than node perturbation, which in turn is faster by a factor of $N$ than weight perturbation. The difference in the rate of convergence is the dimensionality of the noise. Weight perturbation operates by explicit exploration of the entire $MN$-dimensional weight space; only one component of a particular update will be in the direction of the true gradient for a given input, while the other components can be viewed as noise masking that signal. That is, an update can be written as $\Delta W = \langle \Delta W \rangle$ (the 'learning signal', the actual gradient) $+ (\Delta W - \langle \Delta W \rangle)$ (the 'learning noise'), where the average is taken over all values of $\psi$. This learning noise will typically have magnitude $\sqrt{MN}$ larger than the learning signal, and so $MN$ samples are required in order to average it away. Direct gradient descent gives weight updates that are purely signal in this sense; while still occurring in an $MN$-dimensional space, they are by definition exactly in the direction of the gradient for a given input. Thus no exploration of the weight space nor averaging over multiple samples is necessary, and the maximum learning speed is correspondingly greater. Node perturbation is a stochastic algorithm like weight perturbation, but it explores the $M$-dimensional output space rather than the larger weight space; the learning noise is of lower dimension, and correspondingly fewer samples need to be averaged to reveal a learning signal of a given size.

It has previously been argued that the maximum learning rate should scale, not with the dimensionality of the update as shown here, but with the square root of that dimensionality [4]. That claim is based on the fact that the squared magnitude of the update goes as the number of dimensions, and for a given error landscape and position in parameter space, there will be a maximum update size, greater than which instability will result. However, a more quantitative approach is to examine the conditions under which error will decrease, as we have done above. Rather than stopping with the statement that the size of the weight update scales as the square root of the number of dimensions, we have shown that this fact implies that the restriction on convergence scales with the first power of the dimensionality. Numerical simulations of error curves, averaged over many individual trials with online updating, support these conclusions with respect to both the quantitative shapes of the learning curves and the scaling behavior of the conditions on convergence (Fig. 1).

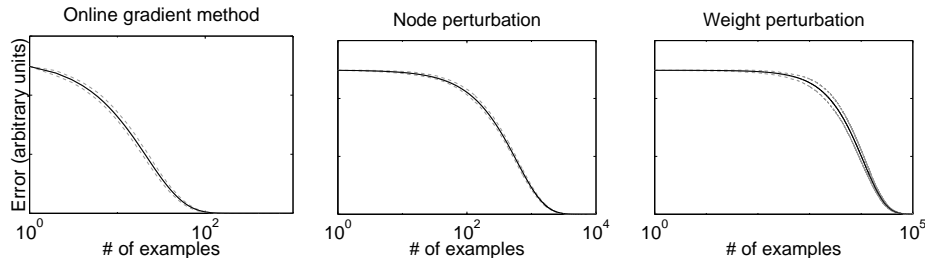

Figure 1: Sample learning curves for the three algorithms applied to a linear feedforward network as described in the text, showing the agreement between theory (black) and experiment (gray). In each case, a network of linear units with $N = 20, M = 25, \sigma = 10^{-3}$, and optimal $\eta$ was trained on successive input examples for the number of iterations shown. 100 such runs were averaged together in each case; the three gray lines show the mean (solid) and standard deviation (dashed) of squared error among those runs.

This scaling result means that, for these stochastic methods, there is no net advantage in speed of training when all degrees of freedom are varied at the same time, compared to when they are varied sequentially, in terms of scaling with $M$ and $N$. For instance, in the case of weight perturbation, varying only one weight at a time would allow the learning rate to be increased by a factor on the order of $MN$; but each of the $MN$ weights would need to be trained in this way, so that the total training time required would scale in the same way as if all were varied at once. (The speed of learning for parallel vs. sequential variation, however, can differ by a *constant* ratio, though we do not pursue this issue here.)

The analysis here describes the behavior in a worst case of sorts, where the objective function and distribution of inputs are isotropic. In the anisotropic case, where the problem is effectively lower-dimensional, the scaling behavior of all three methods can be correspondingly more favorable than that derived here, and the relative performance of the stochastic methods can be better.

The results described in this paper extend at least qualitatively to more complicated networks and architectures. For instance, Fig. 2 shows learning curves that result from applying the three algorithms to a two-layer feedforward network of nonlinear units. All three algorithms give identical learning curves if the learning rate is set small enough; as $\eta$ is increased, the weight perturbation curve fails to converge to low error, while the other two curves continue to match; increasing $\eta$ further leads to the node perturbation curve also failing to converge.

In the above, we have shown that stochastic gradient descent techniques can be expected to scale with increasing network size more poorly than direct ones, in terms of maximum learning rate. This may serve as a caution regarding the size of networks they may usefully be applied to. However, with learning rates in the regime where error converges, equal learning curves in each of the three will follow from equal learning rates, although individual weight updates will typically be considerably different. This is because for correspondingly small adjustments to the weights, only the component parallel to the gradient will have a significant effect on error; orthogonal components will not affect the error to first order. Moreover, node perturbation can have performance comparable to that of direct gradient descent even in training very large networks, so long as the number of output units is small [6]. Thus these stochastic methods may be of considerable utility for training networks in some situations, particularly in reinforcement learning frameworks and those where the gradient of the objective function is difficult or impossible to calculate, for mathematical or practical reasons.

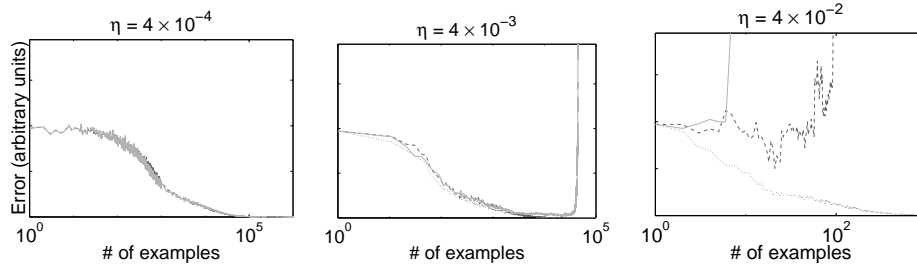

Figure 2: Sample learning curves for the three algorithms applied to a two-layer nonlinear feedforward network (gradient descent, black dotted; node perturbation, dark gray dashed; weight perturbation, light gray solid). The input, hidden, and output layers each had 10 units, whose output was equal to the hyperbolic tangent of their weighted input. Inputs and noises were drawn from the same distributions as in the linear case; $\sigma = 10^{-3}, \eta$ had the value shown for all three algorithms in each panel. In each case, the network was trained on successive input examples for the number of iterations shown; curves show single representative runs. Error was evaluated based on the total squared difference between the output of the network and that of a teacher network with randomly chosen weights; the test error shown was the mean of that for 100 random inputs not used in training.

## Acknowledgments

We thank Ila Fiete and Gert Cauwenberghs for useful discussions and comments. This work was supported in part by a Packard Foundation Fellowship (to H.S. Seung) and NIH grants (GM07484 to MIT and MH60651 to H.S. Seung).

## References

[1] Widrow, B. & Lehr, M. A. 30 years of adaptive neural networks: Perceptron, Madaline, and backpropagation. *Proc. IEEE* **78**(9):1415–1442, 1990.

[2] Jabri, M. & Flower, B. Weight perturbation: an optimal architecture and learning technique for analog VLSI feedforward and recurrent multilayered networks. *IEEE Transactions on Neural Networks* **3**(1):154–157, 1992.

[3] Flower, B. & Jabri, M. Summed weight neuron perturbation: an $\mathcal{O}(n)$ improvement over weight perturbation. In *Advances in Neural Information Processing Systems 5*, San Mateo, CA: Morgan Kaufman Publishers: 212–219, 1993.

[4] Cauwenberghs, G. A fast stochastic error-descent algorithm for supervised learning and optimization. In *Advances in Neural Information Processing Systems 5*, San Mateo, CA: Morgan Kaufman Publishers: 244–251, 1993.

[5] Cauwenberghs, G. An analog VLSI recurrent neural network learning a continuous-time trajectory. *IEEE Transactions on Neural Networks* **7**(2):346–361, 1996.

[6] Fiete, I. Private communication.

[7] Bartlett, P. & Baxter, J. Hebbian synaptic modifications in spiking neurons that learn. Technical report, November 27 1999.

[8] Baldi, P. & Hornik, K. Learning in linear neural networks: a survey. *IEEE Transactions on Neural Networks* **6**(4):837–858, 1995.

[9] Williams, R.J. Simple statistical gradient-following algorithms for connectionist reinforcement learning. *Machine Learning* **8**:229–256, 1992.
